# On Kernel-Target Alignment

**Nello Cristianini**
BIOwulf Technologies
*nello@support-vector.net*

**John Shawe-Taylor**
Royal Holloway, University of London
*john@cs.rhul.ac.uk*

**Andre Elisseeff**
BIOwulf Technologies
*andre@barnhilltechnologies.com*

**Jaz Kandola**
Royal Holloway, University of London
*jaz@cs.rhul.ac.uk*

## Abstract

We introduce the notion of kernel-alignment, a measure of similarity between two kernel functions or between a kernel and a target function. This quantity captures the degree of agreement between a kernel and a given learning task, and has very natural interpretations in machine learning, leading also to simple algorithms for model selection and learning. We analyse its theoretical properties, proving that it is sharply concentrated around its expected value, and we discuss its relation with other standard measures of performance. Finally we describe some of the algorithms that can be obtained within this framework, giving experimental results showing that adapting the kernel to improve alignment on the labelled data significantly increases the alignment on the test set, giving improved classification accuracy. Hence, the approach provides a principled method of performing transduction.

**Keywords:** Kernels, alignment, eigenvectors, eigenvalues, transduction

## 1 Introduction

Kernel based learning algorithms [1] are modular systems formed by a general-purpose learning element and by a problem specific kernel function. It is crucial for the performance of the system that the kernel function somehow fits the learning target, that is that in the feature space the data distribution is somehow correlated to the label distribution. Several results exist showing that generalization takes place only when such correlation exists (nofreelunch; luckiness), and many classic estimators of performance (eg the margin) can be understood as estimating this relation. In other words, selecting a kernel in this class of systems amounts to the classic feature and model selection problems in machine learning.

Measuring the similarity between two kernels, or the degree of agreement between a kernel and a given target function, is hence an important problem both for conceptual and for practical reasons. As an example, it is well known that one can obtain complex kernels by combining or manipulating simpler ones, but how can one predict whether the resulting kernel is better or worse than its components?

What a kernel does is to virtually map data into a feature space so that their relative positions in that space are what matters. The degree of clustering achieved in that space, and the relation between the clusters and the labeling to be learned, should be captured by such an estimator.

Alternatively, one could regard kernels as 'oracles' or 'experts' giving their opinion on whether two given points belong to the same class or not. In this case, the correlation between experts (seen as random variables) should provide an indication of their similarity.

We will argue that - if one were in possess of this information - the ideal kernel for a classification target $y(x)$ would be $K(x, z) = y(x)y(z)$. One way of estimating the extent to which the kernel achieves the right clustering is to compare the sum of the within class distances with the sum of the between class distances. This will correspond to the alignment between the kernel and the ideal kernel $y(x)y(z)$. By measuring the similarity of this kernel with the kernel at hand - on the training set - one can assess the degree of fitness of such kernel. The measure of similarity that we propose, 'kernel alignment' would give in this way a reliable estimate of its expected value, since it is sharply concentrated around its mean.

In this paper we will motivate and introduce the notion of Alignment (Section 2); prove its concentration (Section 3); discuss its implications for the generalisation of a simple classifier (Section 4) and deduce some simple algorithms (Section 5) to optimize it and finally report on some experiments (Section 6).

## 2  Alignment

Given an (unlabelled) sample $S = \{x_1, \ldots, x_m\}$, we use the following inner product between Gram matrices, $\langle K_1, K_2 \rangle_F = \sum_{i,j=1}^m K_1(x_i, x_j)K_2(x_i, x_j)$

**Definition 1 Alignment** *The (empirical) alignment of a kernel $k_1$ with a kernel $k_2$ with respect to the sample $S$ is the quantity*

$$\hat{A}(S, k_1, k_2) = \frac{\langle K_1, K_2 \rangle_F}{\sqrt{\langle K_1, K_1 \rangle_F \langle K_2, K_2 \rangle_F}},$$

*where $K_i$ is the kernel matrix for the sample $S$ using kernel $k_i$.*

This can also be viewed as the cosine of the angle between two bi-dimensional vectors $K_1$ and $K_2$, representing the Gram matrices. If we consider $K_2 = yy'$, where $y$ is the vector of $\{-1, +1\}$ labels for the sample, then

$$\hat{A}(S, K, yy') = \frac{\langle K, yy' \rangle_F}{\sqrt{\langle K, K \rangle_F \langle yy', yy' \rangle_F}} = \frac{\langle K, yy' \rangle_F}{m\sqrt{\langle K, K \rangle_F}}, \text{ since } \langle yy', yy' \rangle_F = m^2$$

We will occasionally omit the arguments $K$ or $y$ when these are understood from the context or when $y$ forms part of the sample. In the next section we will see how this definition provides with a method for selecting kernel parameters and also for combining kernels.

## 3  Concentration

The following theorem shows that the alignment is not too dependent on the training set $S$. This result is expressed in terms of 'concentration'. Concentration means that the probability of an empirical estimate deviating from its mean can be bounded as an exponentially decaying function of that deviation.

This will have a number of implications for the application and optimisation of the alignment. For example if we optimise the alignment on a random sample we can

expect it to remain high on a second sample. Furthermore we will show in the next section that if the expected value of the alignment is high, then there exist functions that generalise well. Hence, the result suggests that we can optimise the alignment on a training set and expect to keep high alignment and hence good performance on a test set. Our experiments will demonstrate that this is indeed the case.

The theorem makes use of the following result due to McDiarmid. Note that $\mathbb{E}_S$ is the expectation operator under the selection of the sample.

**Theorem 2** *(McDiarmid [4]) Let $X_1, \ldots, X_n$ be independent random variables taking values in a set $A$, and assume that $f : A^n \to \mathbb{R}$ satisfies for $1 \le i \le n$*

$$\sup_{x_1,\ldots,x_n,\hat{x}_i} |f(x_1,\ldots,x_n) - f(x_1,\ldots,x_{i-1},\hat{x}_i,x_{i+1},\ldots,x_n)| \le c_i,$$

*then for all $\epsilon > 0$,* $\quad P\{|f(X_1,\ldots,X_n) - \mathbb{E}f(X_1,\ldots,X_n)| \ge \epsilon\} \le 2\exp\left(\dfrac{-2\epsilon^2}{\sum_{i=1}^n c_i^2}\right)$

**Theorem 3** *The sample based estimate of the alignment is concentrated around its expected value. For a kernel with feature vectors of norm 1, we have that*

$$P^m\{S : |\hat{A}(S) - A(y)| \ge \hat{\epsilon}\} \le \delta \quad where \quad \hat{\epsilon} = C(S)\sqrt{8\ln(2/\delta)/m}, \tag{1}$$

*for a non-trivial function $C(S)$ and value $A(y)$.*

**Proof**: Let

$$\hat{A}_1(S) = \frac{1}{m^2}\sum_{i,j=1}^m y_i y_j k(x_i,x_j), \hat{A}_2(S) = \frac{1}{m^2}\sum_{i,j=1}^m k(x_i,x_j)^2, \text{ and } A(y) = \frac{\mathbb{E}_S[\hat{A}_1(S)]}{\sqrt{\mathbb{E}_S[\hat{A}_2(S)]}}.$$

First note that $\hat{A}(S) = \hat{A}_1(S)/\sqrt{\hat{A}_2(S)}$. Define $A_1 = \mathbb{E}_S[\hat{A}_1(S)]$ and $A_2 = \mathbb{E}_S[\hat{A}_2(S)]$. First we make use of McDiarmid's theorem to show that $\hat{A}_i(S)$ are concentrated for $i = 1, 2$. Consider the training set $S' = S \setminus \{(x_i, y_i)\} \cup \{(x'_i, y'_i)\}$. We must bound the difference

$$|\hat{A}_j(S) - \hat{A}_j(S')| \le \frac{1}{m^2}(2(m-1)2) < \frac{4}{m},$$

for $j = 1, 2$. Hence, we have $c_i = 4/m$ for all $i$ and we obtain from an application of McDiarmid's Theorem for $j = 1$ and 2,

$$P^m\{S : |\hat{A}_j(S) - A_j| \ge \epsilon\} \quad \le \quad 2\exp\left(\frac{-\epsilon^2 m}{8}\right)$$

Setting $\epsilon = \sqrt{8\ln(2/\delta)/m}$, the right hand sides are less than or equal to $\delta/2$. Hence, with probability at least $1 - \delta$, we have for $j = 1, 2$ $|\hat{A}_j(S) - A_j| < \epsilon$. But whenever these two inequalities hold, we have

$$\begin{aligned}
\left|\hat{A}(S) - A(y)\right| &= \left|\frac{\hat{A}_1(S)}{\sqrt{\hat{A}_2(S)}} - \frac{A_1}{\sqrt{A_2}}\right| \le \frac{\left|\hat{A}_1(S) - A_1\right|}{\sqrt{\hat{A}_2(S)}} + A_1\left|\frac{1}{\sqrt{\hat{A}_2(S)}} - \frac{1}{\sqrt{A_2}}\right| \\
&\le \frac{\epsilon}{\sqrt{\hat{A}_2(S)}} + \frac{A_1(y)}{\sqrt{\hat{A}_2(S)A_2}\left(\sqrt{A_2} + \sqrt{\hat{A}_2(S)}\right)}|A_2 - \hat{A}_2(S)| \\
&\le \frac{\epsilon}{2(\hat{A}_2(S) - \epsilon)\sqrt{\hat{A}_2(S)}}\left(2\hat{A}_2(S) + \hat{A}_1(S) + \epsilon\right) = C(S)\epsilon. \ \square
\end{aligned}$$

**Remark.** We could also define the true Alignment, based on the input distribution $P$, as follows: given functions $f, g : X^2 \longrightarrow \mathbb{R}$, we define $\langle f, g \rangle_P = \int_{X^2} f(x, z) g(x, z) dP(x) dP(z)$, Then the alignment of a kernel $k_1$ with a kernel $k_2$ is the quantity $A(k_1, k_2) = \frac{\langle k_1, k_2 \rangle_P}{\sqrt{\langle k_1, k_1 \rangle_P \langle k_2, k_2 \rangle_P}}$.

Then it is possible to prove that asymptotically as $m$ tends to infinity the empirical alignment as defined above converges to the true alignment. However if one wants to obtain unbiased convergence it is necessary to slightly modify its definition by removing the diagonal, since for finite samples it biases the expectation by receiving too large a weight. With this modification $A(y)$ in the statement of the theorem becomes the true alignment. We prefer not to pursue this avenue further for simplicity in this short article, we just note that the change is not significant.

# 4    Generalization

In this section we consider the implications of high alignment for the generalisation of a classifier. By generalisation we mean the test error $\text{err}(h) = P(h(x) \neq y)$. Our next observation relates the generalisation of a simple classification function to the value of the alignment. The function we consider is the expected Parzen window estimator $h(x) = \text{sign}(f(x)) = \text{sign}\left(\mathbb{E}_{(x', y')}[y'k(x', x)]\right)$. This corresponds to thresholding a linear function $f$ in the feature space. We will show that if there is high alignment then this function will have good generalisation. Hence, by optimising the alignment we may expect Parzen window estimators to perform well. We will demonstrate that this prediction does indeed hold good in experiments.

**Theorem 4** *Given any $\delta > 0$. With probability $1 - \delta$ over a randomly drawn training set $S$, the generalisation accuracy of the expected Parzen window estimator $h(x) = \text{sign}\left(\mathbb{E}_{(x', y')}[y'k(x', x)]\right)$ is bounded from above by*

$$\text{err}(h(x)) \leq 1 - \hat{A}(S) + \hat{\epsilon} + \left(m\sqrt{\hat{A}_2(S)}\right)^{-1}, \text{ where } \hat{\epsilon} = C(S)\sqrt{\frac{8}{m}\ln\frac{4}{\delta}}.$$

**Proof**: (sketch) We assume throughout that the kernel has been normalised so that $k(x, x) = 1$ for all $x$. First observe that by Theorem 3 with probability greater than $1 - \delta/2$, $|A(y) - \hat{A}(S)| \leq \hat{\epsilon}$. The result will follow if we show that with probability greater than $1 - \delta/2$ the generalisation error of $h_{S \setminus (x_1, y_1)}$ can be upper bounded by $1 - A(y) + \frac{1}{m\sqrt{A_2(S)}}$. Consider the quantity $A(y)$ from Theorem 3.

$$A(y) = \frac{\mathbb{E}_S\left[\frac{1}{m^2}\sum_{ij=1}^m y_i y_j k(x_i, x_j)\right]}{\sqrt{\mathbb{E}_S\left[\frac{1}{m^2}\sum_{ij=1}^m k(x_i, x_j)^2\right]}} = \frac{\mathbb{E}_S\left[\frac{1}{m^2}\sum_{i \neq j}^m y_i y_j k(x_i, x_j)\right] + \frac{1}{m}}{C}$$

where $\quad C = \sqrt{\frac{m-1}{m}\mathbb{E}_{(x,y),(x',y')}[k(x', x)^2] + \frac{1}{m}}$. Observe that

$$\mathbb{E}_S\left[\frac{1}{Cm^2}\sum_{i \neq j}^m y_i y_j k(x_i, x_j)\right] = \mathbb{E}_S\left[\frac{y_1}{Cm}\sum_{i=2}^m y_i k(x_i, x_1)\right] = \mathbb{E}_{(x,y)}\left[y\frac{m-1}{Cm}f(x)\right].$$

But $\quad \left|\frac{m-1}{Cm}f(x)\right| \leq \sqrt{\mathbb{E}_{(x,y)}[y^2]}\sqrt{\frac{(m-1)^2}{C^2 m^2}\mathbb{E}_{(x', y')}[k(x, x')^2]} < 1$

Hence, if $\epsilon = P(f(x) \neq y)$ and $\alpha = P(f(x) = y)$, we have $\mathbb{E}_S\left[\frac{1}{Cm^2}\sum_{i \neq j}^m y_i y_j k(x_i, x_j)\right] \leq 1 \times \alpha + 0 \times \epsilon = \alpha$ and $\epsilon = 1 - \alpha \leq 1 - A(y) + \frac{1}{Cm}$. □

An empirical estimate of the function $f$ would be the Parzen window function. The expected margin of the empirical function is concentrated around the expected margin of the expected Parzen window. Hence, with high probability we can bound the error of $\hat{f}$ in terms of the empirically estimated alignment $\hat{A}(S)$. This is omitted due to lack of space. The concentration of $\hat{f}$ is considered in [3].

## 5 Algorithms

The concentration of the alignment can be directly used for tuning a kernel family to the particular task, or for selecting a kernel from a set, with no need for training. The probability that the level of alignment observed on the training set will be out by more than $\hat{\epsilon}$ from its expectation for one of the kernels is bounded by $\delta$, where $\hat{\epsilon}$ is given by equation (1) for $\epsilon = \sqrt{\frac{8}{m}\left(\ln|N| + \ln\frac{2}{\delta}\right)}$, where $|N|$ is the size of the set from which the kernel has been chosen. In fact we will select from an infinite family of kernels. Providing a uniform bound for such a class would require covering numbers and is beyond the scope of this paper. One of the main consequences of the definition of kernel alignment is in providing a practical criterion for combining kernels. We will justify the intuitively appealing idea that two kernels with a certain alignment with a target that are not aligned to each other, will give rise to a more aligned kernel combination. In particular we have that

$$\hat{A}_{k_1+k_2}(y) = \hat{A}_{k_1}(y)\frac{\|K_1\|_F}{\|K_1 + K_2\|_F} + \hat{A}_{k_2}(y)\frac{\|K_2\|_F}{\|K_1 + K_2\|_F}$$

This shows that if two kernels with equal alignment to a given target $y$ are also completely aligned to each other, then $\|K_1 + K_2\|_F = \|K_1\|_F + \|K_2\|_F$ and the alignment of the combined kernel remains the same. If on the other hand the kernels are not completely aligned, then the alignment of the combined kernel is correspondingly increased.

To illustrate the approach we will take to optimising the kernel, consider a kernel that can be written in the form $k(x, x') = \sum_k \mu_k(y^k(x)y^k(x'))$, where all the $y^k$ are orthogonal with respect to the inner product defined on the training set $S$, $\langle y, y'\rangle_S = \sum_{i=1}^m y_i y_j$. Assume further that one of them $y^t$ is the true label vector. We can now evaluate the alignment as $\hat{A}(y) \approx \mu_t/\sqrt{\sum_k \mu_k^2}$. In terms of the Gram matrix this is written as $K_{ij} = \sum_k \mu_k y_i^k y_j^k$ where $y_i^k$ is the $i$-th label of the $k$-th classification. This special case is approximated by the decomposition into eigenvectors of the kernel matrix $K = \sum \lambda_i v_i v_i'$, where $v'$ denotes the transpose of $v$ and $v_i$ is the $i$-th eigenvector with eigenvalue $\lambda_i$. In other words, the more peaked the spectrum the more aligned (specific) the kernel can be.

If by chance the eigenvector of the largest eigenvalue $\lambda_1$ corresponds to the target labeling, then we will give to that labeling a fraction $\lambda_1/\sqrt{\sum_i \lambda_i^2}$ of the weight that we can allocate to different possible labelings. The larger the emphasis of the kernel on a given target, the higher its alignment.

In the previous subsection we observed that combining non-aligned kernels that are aligned with the target yields a kernel that is more aligned to the target. Consider the base kernels $K_i = v_i v_i'$ where $v_i$ are the eigenvectors of $K$, the kernel matrix for both labeled and unlabeled data. Instead of choosing only the most aligned ones, one could use a linear combination, with the weights proportional to their alignment (to the available labels): $\hat{K} = \sum_i f(\alpha_i)v_i v_i'$ where $\alpha_i$ is the alignment of the kernel $K_i$, and $f(\alpha)$ is a monotonically increasing function (eg. the identity or an exponential). Note that a recombination of these rank 1 kernels was made in so-called latent semantic kernels [2]. The overall alignment of the new kernel with

the labeled data should be increased, and the new kernel matrix is expected also to be more aligned to the unseen test labels (because of the concentration, and the assumption that the split was random).

Moreover, in general one can set up an optimization problem, aimed at finding the optimal $\alpha$, that is the parameters that maximize the alignment of the combined kernel with the available labels. Given $K = \sum_i \alpha_i v_i v_i'$, using the orthonormality of the $v_i$ and that $\langle vv', uu' \rangle_F = \langle v, u \rangle_F^2$, the alignment can be written as

$$\hat{A}(y) = \frac{\langle K, yy' \rangle_F}{m\sqrt{\sum_{ij} \alpha_i \alpha_j \langle v_i v_i', v_j v_j' \rangle_F}} = \frac{\sum_i \alpha_i \langle v_i, y \rangle_F^2}{\sqrt{\langle yy', yy' \rangle_F}\sqrt{\sum_i \alpha_i^2}}.$$

Hence we have the following optimization problem:

$$\text{maximise} \quad W(\alpha) = \sum_i \alpha_i \langle v_i, y \rangle_F^2 - \lambda(\sum_i \alpha_i^2 - 1). \quad (2)$$

Setting derivatives to zero we obtain $\frac{\partial W}{\partial \alpha_i} = \langle v_i, y \rangle_F^2 - \lambda 2\alpha_i = 0$ and hence $\alpha_i \propto \langle v_i, y \rangle_F^2$, giving the overall alignment $\hat{A}(y) = \frac{\sqrt{\sum_i \langle v_i, y \rangle_F^4}}{m}$.

This analysis suggests the following transduction algorithm. Given a partially labelled set of examples optimise its alignment by adapting the full kernel matrix by recombining its rank one eigenmatrices $v_i v_i'$ using the coefficients $\alpha_i$ determined by measuring the alignment between $v_i$ and $y$ on the labelled examples. Our results suggest that we should see a corresponding increase in the alignment on the unlabelled part of the set, and hence a reduction in test error when using a Parzen window estimator. Results of experiments testing these predictions are given in the next section.

## 6  Experiments

We applied the transduction algorithm designed to take advantage of our results by optimizing alignment with the labeled part of the dataset using the spectral method described above. All of the results are averaged over 20 random splits with the standard deviation given in brackets.    Table 1 shows the alignments of the

|          | Train Align   | Test Align    | Train Align   | Test Align    |
|----------|---------------|---------------|---------------|---------------|
| $K_{80}$ | 0.076 (0.007) | 0.092 (0.029) | 0.207 (0.020) | 0.240 (0.083) |
| $G_{80}$ | 0.228 (0.012) | 0.219 (0.041) | 0.240 (0.016) | 0.257 (0.059) |
| $K_{50}$ | 0.075 (0.016) | 0.084 (0.017) | 0.210 (0.031) | 0.216 (0.033) |
| $G_{50}$ | 0.242 (0.023) | 0.181 (0.043) | 0.257 (0.023) | 0.202 (0.015) |
| $K_{20}$ | 0.072 (0.022) | 0.081 (0.006) | 0.227 (0.057) | 0.210 (0.015) |
| $G_{20}$ | 0.273 (0.037) | 0.034 (0.046) | 0.326 (0.023) | 0.118 (0.017) |

Table 1: Mean and associated standard deviation alignment values using a linear kernel on the Breast (left two columns) and Ionosphere (right two columns).

Gram matrices to the label matrix for different sizes of training set. The index indicates the percentage of training points. The $K$ matrices are before adaptation, while the $G$ matrices are after optimisation of the alignment using equation (2). The results on the left are for Breast Cancer data using a linear kernel, while the results on the right are for Ionosphere data.

The left two columns of Table 2 shows the alignment values for Breast Cancer data using a Gaussian kernel together with the performance of an SVM classifier trained

|          | Train Align   | Test Align    | SVM Error     | Breast        | Ionosphere    |
|----------|---------------|---------------|---------------|---------------|---------------|
| $K_{80}$ | 0.263 (0.011) | 0.242 (0.041) | 0.160 (0.023) | 0.639 (0.03)  | 0.293 (0.06)  |
| $G_{80}$ | 0.448 (0.012) | 0.433 (0.047) | 0.158 (0.027) | 0.196 (0.03)  | 0.261 (0.07)  |
| $K_{50}$ | 0.269 (0.029) | 0.249 (0.026) | 0.151 (0.013) | 0.644 (0.02)  | 0.307 (0.03)  |
| $G_{50}$ | 0.453 (0.026) | 0.437 (0.026) | 0.146 (0.015) | 0.195 (0.02)  | 0.261 (0.03)  |
| $K_{20}$ | 0.251 (0.048) | 0.260 (0.012) | 0.143 (0.007) | 0.648 (0.01)  | 0.312 (0.01)  |
| $G_{20}$ | 0.448 (0.055) | 0.441 (0.014) | 0.144 (0.017) | 0.256 (0.04)  | 0.322 (0.04)  |

Table 2: Breast alignment (cols 1,2) and SVM error for a Gaussian kernel (sigma = 6) (col 3), Parzen window error for Breast (col 4) and Ionosphere (col 5)

with the given gram matrix in the third column. The right two columns show the performance of the Parzen window classifier on the test set for Breast linear kernel (left column) and Ionosphere (right column).

The results clearly show that optimising the alignment on the training set does indeed increase its value in all but one case by more than the sum of the standard deviations. Furthermore, as predicted by the concentration this improvement is maintained in the alignment measured on the test set with both linear and Gaussian kernels in all but one case (20% train with the linear kernel). The results for Ionosphere are less conclusive. Again as predicted by the theory the larger the alignment the better the performance that is obtained using the Parzen window estimator. The results of applying an SVM to the Breast Cancer data using a Gaussian kernel show a very slight improvement in the test error for both 80% and 50% training sets.

# 7 Conclusions

We have introduced a measure of performance of a kernel machine that is much easier to analyse than standard measures (eg the margin) and that provides much simpler algorithms. We have discussed its statistical and geometrical properties, demonstrating that it is a well motivated and formally useful quantity.

By identifying that the ideal kernel matrix has a structure of the type $yy'$, we have been able to transform a measure of similarity between kernels into a measure of fitness of a given kernel. The ease and reliability with which this quantity can be estimated using only training set information prior to training makes it an ideal tool for practical model selection. We have given preliminary experimental results that largely confirm the theoretical analysis and augur well for the use of this tool in more sophisticated model (kernel) selection applications.

# References

[1] N. Cristianini and J. Shawe-Taylor. *An Introduction to Support Vector Machines*. Cambridge University Press, 2000. See also the web site **www.support-vector.net**.

[2] Nello Cristianini, Huma Lodhi, and John Shawe-Taylor. Latent semantic kernels for feature selection. Technical Report NC-TR-00-080, NeuroCOLT Working Group, http://www.neurocolt.org, 2000.

[3] L. Devroye, L. Györfi, and G. Lugosi. *A Probabilistic Theory of Pattern Recognition*. Number 31 in Applications of mathematics. Springer, 1996.

[4] C. McDiarmid. On the method of bounded differences. In *Surveys in Combinatorics 1989*, pages 148–188. Cambridge University Press, 1989.
